# Minimizing Uncertainty in Pipelines[*]

**Nilesh Dalvi**
Facebook, Inc.
nileshd@fb.com

**Aditya Parameswaran**
Stanford University
adityagp@cs.stanford.edu

**Vibhor Rastogi**
Google, Inc.
vibhor.rastogi@gmail.com

## Abstract

In this paper, we consider the problem of debugging large pipelines by human labeling. We represent the execution of a pipeline using a directed acyclic graph of AND and OR nodes, where each node represents a data item produced by some operator in the pipeline. We assume that each operator assigns a confidence to each of its output data. We want to reduce the uncertainty in the output by issuing queries to a human, where a query consists of checking if a given data item is correct. In this paper, we consider the problem of asking the optimal set of queries to minimize the resulting output uncertainty. We perform a detailed evaluation of the complexity of the problem for various classes of graphs. We give efficient algorithms for the problem for trees, and show that, for a general dag, the problem is intractable.

## 1 Introduction

In this paper, we consider the problem of debugging pipelines consisting of a set of data processing operators. There is a growing interest in building various web-scale automatic information extraction pipelines [9, 10, 14, 7], with operators such as clustering, extraction, classification, and deduplication. The operators are often based on machine learned models, and they associate confidences with the data items they produce. At the end, we want to resolve the uncertainties of the final output tuples, i.e., figure out which of them are correct and which are incorrect.

.5Given a fixed labeling budget, we can only inspect a subset of the output tuples. However, the output uncertainties are highly correlated since different tuples share their lineage. Thus, inspecting a tuple also gives us information about the correctness of other tuples. In this paper, we consider the following interesting and non-trivial problem : *given a budget of k tuples, choose the k tuples to inspect that minimize the total uncertainty in the output*. We will formalize the notion of a data pipeline and uncertainty in Section 2. Here, we illustrate the problem using an example.

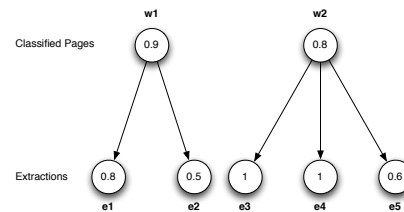

Figure 1: Pipeline Example

**Example 1.1.** *Consider a simple hypothetical pipeline for extracting computer scientists from the Web that consists of two operators: a classifier that takes a webpage and determines if it is a page about computer science, and a name extractor that extracts names from a given webpage. Fig. 1 shows an execution of this pipeline. There are two webpages, $w_1$ and $w_2$, output by the classifier. The extractor extracts entities $e_1$ and $e_2$ from $w_1$ and $e_3$, $e_4$ and $e_5$ from $w_2$. Each operator also gives a confidence with its output. In Fig. 1, the classifier attaches a probability of $0.9$ and $0.8$ to pages $w_1$ and $w_2$. Similarly, the extractor attaches a probability to each of the extractions $e_1$ to $e_5$. The probability that an operator attaches to a tuple is conditioned on the correctness of its input. Thus, the final probability of $e_1$ is $0.8 \times 0.9 = 0.72$. Similarly, the final probabilities of $e_2$ to $e_5$ are $0.45$, $0.8$, $0.8$ and $0.48$ respectively. Note that the uncertainties are correlated, e.g., $e_3$ and $e_4$ are either both correct or both incorrect. We want to choose k tuples to inspect that minimize the total output uncertainty.*

---

[*]This work was partly done when the authors were employed at Yahoo! Research.

| Graph | BEST-1 | INCR | BEST-K |
|---|---|---|---|
| TREE(2) | $O(n)$ | $O(n)$ or $O(\log n)+O(n\log n)$ preprocessing | OPEN (Weakly PTIME) 2-approximate[†]: $O(n\log n)$ |
| TREE | $O(n)$ | $O(n)$ | OPEN ($O(n^{k+1})$) |
| DAG(2,$\wedge$) or DAG($\wedge$) | $O(n^3)$ | PP-hard (Probabilistic Polynomial) Hard to Approximate | PP-Hard Hard to Approximate |
| DAG(2,$\vee$) | $O(n^3)$ | PP-hard, Hard to Approximate | PP-Hard, Hard to Approximate |
| DAG($\vee$) | PP-Hard | PP-Hard, Hard to Approximate | |
| DAG | | PP-Hard, Hard to Approximate | |

Table 1: Summary of Results; [†]Twice the number of queries to achieve same objective as optimal

If all the data items were independent, we would have queried the most uncertain items, i.e. having probability closest to 1/2. However, in presence of correlations between the output tuples, the problem becomes non-trivial. For instance, let us revisit the first example with $k = 1$, i.e., we can inspect one tuple. Of the 5 output tuples, $e_5$ is the most uncertain, since its probability 0.48 is closest to 1/2. However, one might argue that $e_3$ (or $e_4$) is more informative item to query, since the extractor has a full confidence on $e_3$. Thus, $e_3$ is correct iff $w_2$ is correct (i.e. the classifier was correct on $w_2$). Resolving $e_3$ completely resolves the uncertainty in $w_2$, which, in turn, completely resolves the uncertainty in $e_4$ and reduces the uncertainty in $e_5$. The argument holds even when the extractor confidence in $e_3$ is less than 1 but still very high. In general, one can also query intermediate nodes in addition to the output tuples, and choosing the best node is non-trivial.

In this paper, we consider the general setting of a data pipeline given by a directed acyclic graph that can capture both the motivating scenarios. We define a measure of total uncertainty of the final output based on how close the probabilities are to either 0 or 1. We give efficient algorithms to find the set of data items to query that minimizes the total uncertainty of the output, both under interactive and batch settings.

## 1.1 Related Work

Our problem is an instance of active learning [27, 13, 12, 17, 2, 15, 5, 4, 3] since our goal is to infer probability values of the nodes being true in the DAG, by asking for tags of example nodes. The metric that we use is similar to the square loss metric. However, our problem has salient differences. Unlike traditional active learning where we want to learn the underlying probabilistic model from iid samples, in our problem, we already know the underlying model and want to gain information about non-iid items with known correlations. This makes our setting novel and interesting.

Our DAG structure is a special case of Bayesian networks [6]. A lot is known about general bayes-net inference [21]. For instance, MAP inference given evidence is NP$^{PP}$-complete [24] (approximate inference is NP-complete [1]), inferring whether the probability of a set of variables taking a certain values given evidence about others is $> 0$ is NP-complete [8], is $> t$ is PP-complete [22], while finding its values is #P-complete [26]. However, these results do not apply to our problem setting. In our setting, we are given a set of non-iid items whose correlatations are given by a Bayesian network with known structure and probabilities. We want to choose a subset of items, conditioned on which, the uncertinty of the remaining items is minimized.

Our work is closely related to the field of active diagnosis [28, 19, 20], where the goal is to infer the state of unknown nodes in a network by selecting suitable "test probes". From this field, the most closely related work is that by Krause and Guestrin [19], which considers minimization of uncertainty in a Bayesian network. In that work, the goal is to identify a subset of variables in a graphical model that would minimize the joint uncertainty of a target set of variables. Their primary result is a proof of submodularity under suitable independence assumptions on the graphical model which is then used to derive an approximation algorithm to pick variables. In our problem setting submodularity does not hold, and hence the techniques do not apply. On the other hand, since our graphical model has a specific AND/OR structure, we are able to concretely study the complexity of the algorithms. Our work is also related to the work on graph search [23], where the goal is to identify hidden nodes while asking questions to humans. Since the target applications are different, the underlying model in that work is less general.

## 2 Problem Statement

**Execution Graph:** Let $G$ be a directed acyclic graph (dag), where each node $n$ in $G$ has a label from the set $\{\wedge, \vee\}$ and a probability $p(n)$. We call such a graph a *probabilistic and-or dag*. We denote

the class of such graphs as DAG. We represent the results of an execution of a pipeline of operators using a probabilistic and-or dag.

The semantics of $G \in$ DAG is as follows. Each node in $G$ represents a data item. The parents of a node $n$, i.e. the set of nodes having an outgoing edge to $n$, denote the set of data items which were input to the instance of the operator that produced $n$. We use $parent(n)$ to denote the parents of $n$. The probability $p(n)$ denotes the probability that the data item $n$ is correct conditioned on $parent(n)$ being correct. If $n$ has label $\wedge$, then it requires all the parents to be correct. If $n$ has label $\vee$, it requires at least one parent to be correct. We further assume that, conditioned on the parents being correct, nodes are correct independently.

To state the semantics formally, we associate a set of independent Boolean random variables $X(n)$ for each node $n$ in $G$ with probability $p(n)$. We also associate another set of random variables $Y(n)$, which denotes whether the result at node $n$ is correct (unconditionally). For a $\wedge$ node, $Y(n)$ is defined as: $Y(n) = X(n) \wedge \bigwedge_{m \in parent(n)} Y(m)$. For a $\vee$ node, $Y(n)$ is defined as: $Y(n) = X(n) \wedge \bigvee_{m \in parent(n)} Y(m)$.

When $G$ is a tree, i.e., all nodes have a single parent, the labels of nodes do not have any effect, since $Y(n)$ is the same for both $\wedge$ and $\vee$ nodes. In this case, we simply treat $G$ as an unlabeled tree. For instance, Figure 1 denotes the (unlabeled) tree for the pipeline given in Example 1.1. Thus probabilistic and-or dags provide a powerful formalism to capture data pipelines in practice such as the one in Example 1.1.

**Output Uncertainty:** Let $L$ denote the set of leaves of $G$, which represent the final output of the pipeline. We want all the final probabilities of $L$ to be close to either 0 or 1, as the closer the probability to 1/2, the more uncertain the correctness of the given node is. Let $f(p)$ denote some measure of uncertainty of a random variable as a function of its probability $p$. Then, we define the total output uncertainty of the DAG as

$$I = \sum_{n \in L} f(\Pr(Y(n))) \tag{1}$$

Our results continue to hold when different $n \in L$ are weighted differently, i.e., we use a weighted version of Eq. (1). We describe this simple extension in the extended technical report [11].

Now, our goal is to query a set of nodes $Q$ that minimize the expected total output uncertainty conditioned on observing $Q$. We define this as follows. Let $Q = \{l_1, l_2, \cdots, l_k\}$ be a set of nodes. Given $\mathbf{v} = \{v_1, \cdots, v_k\} \in \{0,1\}^k$, we use $Q = \mathbf{v}$ to denote the event $Y(l_i) = v_i$ for each $i$. Then, define

$$I(Q) = \sum_{\mathbf{v} \in \{0,1\}^k} \Pr(Q = \mathbf{v}) \sum_{n \in L} f(\Pr(Y(n) \mid Q = \mathbf{v})) \tag{2}$$

The most basic version of our problem is following.

---
**Problem 1** (Best-1). *Given a $G \in$ DAG, find the node $q$ that minimizes the expected uncertainty $I(\{q\})$.*

---

A more challenging question is the following:

---
**Problem 2** (Best-k). *Given a $G \in$ DAG, find the set of nodes $Q$ of size $k$ that minimizes $I(Q)$.*

---

In addition to this, we also consider the incremental version of the problem defined as follows. Suppose we have already issued a set of queries $Q_0$ and obtained a vector $\mathbf{v_0}$ of their correctness values. Given a new set of queries, we define the conditioned uncertainty as $I(Q \mid Q_0 = \mathbf{v_0}) = \sum_{\mathbf{v}} \Pr(Q = \mathbf{v} \mid Q_0 = \mathbf{v_0}) \sum_{n \in L} f(\Pr(Y(n) \mid Q = \mathbf{v} \wedge Q_0 = \mathbf{v_0}))$. We also pose the following question:

---
**Problem 3** (Incr). *Given a $G \in$ DAG, and a set of already issued queries $Q_0$ with answer $\mathbf{v_0}$, find the best node $q$ to query next that minimizes $I(\{q\} \mid Q_0 = \mathbf{v_0})$.*

---

In this work, we use the uncertainty metric given by

$$f(p) = p(1-p) \tag{3}$$

Thus, $f(p)$ is minimized when $p$ is either 0 or 1, and is maximum at $p = 1/2$. Note that $f(p) = 1/4 - (1/2 - p)^2$. Hence, minimizing $f(p)$ is equivalent to maximizing the squares of differences of probabilities with 1/2. We call this the $\mathbf{L}^2$ metric. There are other reasonable choices for the uncertainty metric, e.g. $\mathbf{L}^1$ or entropy. The actual choice of uncertainty metrics is not important for our application. In the technical report [11], we show that using any of these different metrics, the resulting solutions are "similar" to each other.

Our uncertainty objective function can be shown to satisfy some desirable properties, such as:

**Theorem 2.1** (Information Never Hurts). *For any sets of queries $Q_1$, $Q_2$, $I(Q_1) \geq I(Q_1 \cup Q_2)$*

Thus, expected uncertainty cannot increase with more queries. Further, the objective function $I$ is neither sub-modular nor super-modular. These results continue to hold when $f$ is replaced with other metrics (Sec. 6). Lastly, for the rest of the paper, we will assume that the query nodes $Q$ are selected from only among the leaves of $G$. This is only to simplify the presentation. There is a simple reduction of the general problem to this problem, where we attach a new leaf node to every internal node, and set their probabilities to 1. Thus, for any internal node, we can equivalently query the corresponding leaf node (we will need to use the weighted form of the Eq. (1), described in the extended technical report [11], to ensure that new leaf nodes have weight 0 in the objective function.)

## 3  Summary of main results

We first define class of probabilistic and-or dags. Let $\textsc{Dag}(\wedge)$ and $\textsc{Dag}(\vee)$ denote the subclasses of $\textsc{Dag}$ where all the node labels are $\wedge$ and $\vee$ respectively. Let $\textsc{Dag}(2,\wedge)$ and $\textsc{Dag}(2,\vee)$ denote the subclasses where the dags are further restricted to depth 2. (We define the depth to be the number of nodes in the longest root to leaf directed path in the dag.) Similarly, we define the class $\textsc{Tree}$ where the dag is restricted to a tree, and $\textsc{Tree}(d)$, consisting of depth-$d$ trees. For trees, since each node has a single parent, the labels of the nodes do not matter.

We start by defining relationships between expressibility of each of these classes. Given any $D_1, D_2 \in \textsc{Dag}$, we say that $D_1 \equiv D_2$ if they have the same number of leaves, and define the same joint probability distribution on the set of their leaves. Given two classes of dags $\mathscr{C}_1$ and $\mathscr{C}_2$, we say $\mathscr{C}_1 \subset \mathscr{C}_2$ if for all $D_1 \in \mathscr{C}_1$, there is a $D_2 \in \mathscr{C}_2$ s.t. $D_2$ is polynomial in the size of $D_1$ and $D_1 \equiv D_2$.

**Theorem 3.1.** *The following relationships exist between different classes:*

$$\textsc{Tree}(2) \subset \textsc{Tree} \subset \textsc{Dag}(2,\wedge) = \textsc{Dag}(\wedge) \subset \textsc{Dag}(2,\vee) \subset \textsc{Dag}(\vee) \subset \textsc{Dag}$$

Table 1 shows the complexity of the three problems as defined in the previous section, for different classes of graphs. The parameter $n$ is the number of nodes in the graph. While the problems are tractable, and in fact efficient, for trees, they become hard for general dags. Here, PP denotes the complexity class of probabilistic polynomial time algorithms. Unless P = NP, there are no PTIME algorithms for PP-hard problems. Further, for some of the problems, we can show that they cannot be approximated within a factor of $2^{n^{1-\varepsilon}}$ for any positive constant $\varepsilon$ in PTIME.

## 4  Best-1 Problem

We start with the most basic problem: given a probabilistic $\textsc{Dag}$ $G$, find the node to query that minimizes the resulting uncertainty. We first provide PTIME algorithms for $\textsc{Tree}(2), \textsc{Tree}, \textsc{Dag}(\wedge)$, and $\textsc{Dag}(2,\vee)$ (Recall that as we saw earlier, $\textsc{Dag}(2,\vee)$ subsumes $\textsc{Dag}(\wedge)$.) Subsequently, we show that finding the best node to query is intractable for $\textsc{Dag}(\vee)$ of depth greater than 2, and is thus intractable for $\textsc{Dag}$ as well. For $\textsc{Tree}$ and $\textsc{Dag}(\wedge)$, the expression for $Y(n)$ can be rewritten as the following: $Y(n) = \bigwedge_{m \in anc(n)} X(m)$, where $anc(n)$ denotes the set of ancestors of $n$, i.e., those nodes that have a directed path to $n$, including $n$ itself. This "unrolled" formulation will allow us to compute the probabilities $Y(x) = 1$ easily.

### 4.1  $\textsc{Tree}(2)$

Consider a simple tree graph $G$ with root $r$, having $p(r) = p_r$, and having children $l_1, \cdots, l_n$ with $p(l_i) = p_i$. Given a node $x$, let $e_x$ denote the event $Y(x) = 1$, and $\overline{e_x}$ denote the event that $Y(x) = 0$. We want to find the leaf $q$ that minimizes $I(\{q\})$, where:

$$I(\{q\}) = \sum_{l \in L} \Pr(e_q) f(\Pr(e_l \mid e_q)) + \Pr(\overline{e_q}) f(\Pr(e_l \mid \overline{e_q})) \qquad (4)$$

By a slight abuse of notation, we will use $I(q)$ to denote the quantity $I(\{q\})$. It is easy to see the following (let $l \neq q$):

$$\Pr(e_q) = p_r p_q, \qquad \Pr(e_l \mid e_q) = p_l, \qquad \Pr(e_l \mid \overline{e_q}) = p_r p_l (1 - p_q)/(1 - p_r p_q)$$

Substituting these expressions back in Eq. (4), and assuming $f(p) = p(1 - p)$, we get the following:

$$I(q) = \sum_{l \in L, l \neq q} p_r p_q p_l (1 - p_l) + p_r p_l (1 - p_q)(1 - p_r p_l (1 - p_q)/(1 - p_r p_q))$$

We observe that it is of the form

$$F_0(p_q, p_r) + F_1(p_q, p_r) \sum_l p_l + F_2(p_q, p_r) \sum_l p_l^2 \tag{5}$$

where $F_0, F_1, F_2$ are small rational polynomials over $p_r$ and $p_q$. This immediately gives us a linear time algorithm to pick the best $q$. We first compute $\sum_l p_l$ and $\sum_l p_l^2$, and then compute the objective function for all $q$ in linear time.

Now we consider the case when $G$ is any general tree with the set of leaves $L$. Recall that $e_x$ is the event that denotes $Y(x) = 1$. Denote the probability $\Pr(e_x)$ by $P_x$. Thus, $P_x$ is the product of $p(y)$ over all nodes $y$ that are the ancestors of $x$ (including $x$ itself). Given nodes $x$ and $y$, let $lca(x, y)$ denote the least common ancestor of $x$ and $y$. Our objective is to find $q \in L$ that minimizes Eq. (4). The following is immediate:

$$\Pr(e_q) = P_q \qquad \Pr(e_l \mid e_q) = \frac{P_l}{P_{lca(l,q)}} \qquad \Pr(e_l \mid \overline{e_q}) = \frac{P_l(1 - P_q/P_{lca(l,q)})}{1 - P_q}$$

However, if we directly plug this in Eq.(4), we don't get a simple form analogous to Eq.(5). Instead, we group all the leaves into equivalence classes based on their lowest common ancestor with $q$ as shown in Fig. 2.

Let $a_1, \cdots, a_d$ be the set of ancestors of $q$. Consider all leaves in the set $L_i$ such that their lowest common ancestor with $q$ is $a_i$. Given a node $x$, let $S(x)$ denote the sum of $P_l^2$ over all leaves $l$ reachable from $x$. If we sum Eq. (4) over all leaves in $L_i$, we get the following expression:

$$-(S(a_i) - S(a_{i-1})) \frac{(P_q + P_{a_i}^2 - 2P_q P_{a_i})}{P_{a_i}^2(1 - P_q)} + \sum_{l \in L_i} P_l$$

Define $\Delta_1(a_i) = S(a_i) - S(a_{i-1})$ and $\Delta_2(a_i) = (S(a_i) - S(a_{i-1})) \frac{1 - 2P_{a_i}}{P_{a_i}^2}$. We can write the above expression as:

$$-\frac{1}{1 - P_q} \Delta_1(a_i) - \frac{P_q}{1 - P_q} \Delta_2(a_i) + \sum_{l \in L_i} P_l$$

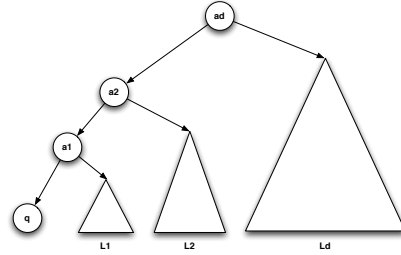

Figure 2: Equivalence Classes of Leaves

Summing these terms over all the ancestors of $q$, we get

$$I(q) = -\frac{1}{1 - P_q} \sum_{a \in anc(q)} \Delta_1(a) - \frac{P_q}{1 - P_q} \sum_{a \in anc(q)} \Delta_2(a) + \sum_{l \in L} P_l$$

## 4.2 TREE

Our main observation is that we can compute $I(q)$ for all leaves together in time linear in the size of $G$. First, using a single top-down dynamic programming over the tree, we can compute $P_x$ for all nodes $x$. Next, using a single bottom-up dynamic programming over $G$, we can compute $S(x)$ for all nodes $x$. In the third step, we compute $\Delta_1(x)$ and $\Delta_2(x)$ for all nodes in the tree. In the fourth step, we compute $\sum_{a \in anc(x)} \Delta_i(x)$ for all nodes in the graph using another top-down dynamic programming. Finally, we scan all the leaves and compute the objective function using the above expression. Each of the 5 steps runs in time linear in the size of the graph. Thus, we have

**Theorem 4.1.** *Given a tree $G$ with $n$ nodes, we can compute the node $q$ that minimizes $I(q)$ is time $O(n)$.*

## 4.3 DAG$(2, \vee)$

We now consider DAG$(2, \vee)$. As before, we want to find the best node $q$ that minimizes $I(q)$ as given by Eq. (4). However, the expressions for probabilities $\Pr(e_q)$ and $\Pr(e_l \mid e_q)$ are more complex for DAG$(2, \vee)$. First, note that $P_l$, i.e., the probability that $\Pr(Y(l) = 1)$ is computed as follows: $P_l = p(l) \times \left(1 - \prod_{x \in parent(l)} (1 - p(x))\right)$. The probability that at least one of the shared ancestors of $l$ and $q$ are true is: $P_{l,q} = 1 - \prod_{x \in parent(l) \cap parent(q)} (1 - p(x))$. And the probability that one of the unique ancestors of $l$ is true is: $P_{l \setminus q} = 1 - \prod_{x \in parent(l) \setminus parent(q)} (1 - p(x))$ Then, the following are

immediate:

$$\Pr(e_q) = P_q$$

$$\Pr(e_q \mid e_l) = \frac{p(l) \cdot p(q) \cdot (P_{l,q} + (1 - P_{l,q}) \cdot P_{l\setminus q} \cdot P_{q\setminus l})}{P_l}$$

$$\Pr(e_q \mid \overline{e_l}) = \frac{P_q \cdot (1 - p(l)) + p(l) \cdot p(q) \cdot (1 - P_{l,q}) \cdot (1 - P_{l\setminus q}) \cdot P_{q\setminus l}}{1 - P_l}$$

Note that $P_l, P_{l,q}, P_{l\setminus q}$ can be computed for one $l, q$ pair in time $O(n)$ and thus for all $l, q$ in time $O(n^3)$. Subsequently, finding the best candidate node would require $O(n^2)$ time, giving us an overall $O(n^3)$ algorithm to find the best node.

**Theorem 4.2.** *Given $G \in \mathrm{DAG}(2, \vee)$ with $n$ nodes, we can compute $q$ that minimizes $I(q)$ is time $O(n^3)$.*

Since every $\mathrm{DAG}(\wedge)$ can be converted into to one in $\mathrm{DAG}(2, \vee)$ in $O(n^3)$ (see [11]), we get:

**Theorem 4.3.** *Given $G \in \mathrm{DAG}(\wedge)$ with $n$ nodes, we can compute $q$ that minimizes $I(q)$ is time $O(n^3)$.*

### 4.4  $\mathrm{DAG}(\vee)$

**Theorem 4.4** (Hardness of Best-1 for $\mathrm{DAG}(\vee)$). *The best-1 problem for $\mathrm{DAG}(\vee)$ is PP-Hard.*

We use a reduction from the decision version of the #P-Hard *monotone-partitioned-2-DNF problem* [25]. The proof can be found in the extended technical report [11]. Thus, incremental and best-k problems for $\mathrm{DAG}(\vee)$ are PP-Hard as well. As a corollary from Theorem 3.1 we have:

**Theorem 4.5** (Hardness of Best-1 for $\mathrm{DAG}$). *The best-1 problem for $\mathrm{DAG}$ is PP-Hard.*

This result immediately shows us that the incremental and best-k problems for $\mathrm{DAG}$ are PP-Hard. However, we can actually prove a stronger result for $\mathrm{DAG}$, i.e., that they are hard to approximate. We use a weakly parsimonious reduction from the #P-Hard *monotone-CNF problem*. Note that unlike the partitioned-2-DNF problem (used for the reduction above), which admits a FPRAS (Fully Polynomial Randomized Approximation Scheme) [18], monotone-CNF is known to be hard to approximate [26]. In our proof, we use the fact that repeated applications of an approximation algorithm for best-1 for $\mathrm{DAG}$ would lead to an approximation algorithm for monotone-CNF, which is known to be hard to approximate. This result is shown in the extended version [11].

**Theorem 4.6** (Inapproximability for $\mathrm{DAG}$). *The best-1 problem for $\mathrm{DAG}$ is hard to approximate.*

## 5  Incremental Node Selection

In this section, we consider the problem of picking the next best node to query after a set of nodes $Q_0$ have already been queried. We let vector $\mathbf{v_0}$ reflect their correctness values. We next pick a leaf node $q$ that minimizes $I(\{q\} \mid Q_0 = \mathbf{v_0})$. Again, by slightly abusing notation, we will write the expression simply as $I(q \mid Q_0 = \mathbf{v_0})$.

In this section, we first consider $\mathrm{TREE}(2)$ and $\mathrm{TREE}$. Recall from the previous section that the incremental problem is intractable for $\mathrm{DAG}(\vee)$. Here, we prove that incremental picking is intractable for $\mathrm{DAG}(\wedge)$ itself.

### 5.1  $\mathrm{TREE}$

We want to extend our analysis of Sec. 4 by replacing $\Pr(e_x)$ by $\Pr(e_x \mid Q_0 = \mathbf{v_0})$ and $\Pr(e_x \mid e_y)$ by $\Pr(e_x \mid e_y \wedge Q_0 = \mathbf{v_0})$. We will show that, conditioned on $Q_0 = \mathbf{v_0}$, the resulting probability distribution of the leaves can again be represented using a tree. The new tree is constructed as follows.

Given $Q_0 = \mathbf{v_0}$, apply a sequence of transformations to $G \in \mathrm{TREE}$, one for each $q_0 \in Q_0$. Suppose the value of $q_0 = 1$. Then, for each ancestor $a$ of $q_0$ including itself, set $p(a) = 1$. If $q_0 = 0$, then for each ancestor $a$ including itself, change its $p(a)$ to $p(a)\frac{1 - P_{q_0}/P_a}{1 - P_{q_0}}$. Let all other probabilities remain the same.

**Theorem 5.1.** *Let $G'$ be the tree as defined above. Then, $I(q \mid Q_0 = \mathbf{v_0})$ on $G$ is equal to $I(q)$ on $G'$.*

Thus, after each query, we can incorporate the new evidence by updating the probabilities of all the nodes along the path from the query node to the root. Thus, finding the next best node to query can still be computed in linear time.

## 5.2 TREE(2)

For $G \in$ TREE(2), the above algorithm results in the following tree transformation. If a leaf $q$ is queried, and the result is 1, then $p(r)$ and $p(q)$ are set to 1. If the result is 0, $p(q)$ is set to 0 and $p(r)$ is set to $\frac{p_r(1-p_q)}{1-p_r p_q}$.

Instead of using Eq. (5) to compute the next best in linear time, we can devise a more efficient scheme. Suppose we are given all the leaf probabilities in sorted order (or if we sort them initially). Then, we can subsequently compute the leaf $q$ that minimizes Eq. (5) in $O(\log n)$ time: Consider the rational polynomials $F_0, F_1$ and $F_2$. For a fixed $p_r$, $\sum_l p_l$, and $\sum_l p_l^2$, this expression can be treated as a rational polynomial in a single variable $p_q$. If we take the derivative, the numerator is a quartic in $p_q$. Thus, it can have at most four roots. We can find the roots of a quartic using Ferrari's approach in constant time [16]. Using 4 binary searches, we can find the two $p_q$ closest to each of these roots (giving us 8 candidates for $p_q$, plus two more which are the smallest and the largest $p_q$), and evaluate $I(q)$ for each of those 10 candidates. Thus, finding the best $q$ takes $O(\log n)$ time.

Now, given each new evidence (i.e., the answer to each subsequent query), we can update the $p_r$ probability and the sum $\sum_l p_l^2$ in constant time. Given the new polynomial, we can find the new set of roots, and using the same technique as above, find the next best $q$ in $O(\log n)$ time.

**Theorem 5.2.** *If the p values of the leaf nodes are provided in sorted order, then, for a Depth-2 tree, the next best node to query can be computed in $O(\log n)$.*

## 5.3 DAG($\wedge$)

For DAG($\wedge$), while we can pick the best-1 node in $O(n^3)$ time, we have the surprising result that the problem of picking subsequent nodes become intractable. The intuition is that unlike trees, after conditioning on a query node, the resulting distribution can no longer be represented using another dag. In particular, we show that given a set $S$ of queried nodes, the problem of finding the next best node is intractable in the size of $S$. We use a reduction from the *monotone-2-CNF problem*.

**Theorem 5.3** (PP-Hardness of Incr. for DAG($\wedge$)). *The incremental problem in DAG($\wedge$) is PP-Hard.*

Our reduction, shown in in the extended technical report [11], is a weakly parsimonious reduction involving monotone-2-CNF, which is known to be hard to approximate, thus we have the following result:

**Theorem 5.4** (Inapproximability for DAG($\wedge$)). *The Incremental problem for DAG($\wedge$) is hard to approximate.*

The above result, along with Theorem 3.1, implies that DAG($2, \vee$) is also PP-Hard.

# 6 Best-K

In this section, we consider the problem of picking the best $k$ nodes to minimize uncertainty. Krause et al. [19] give a $\log n$ approximation algorithm for a similar problem under the conditions of *super-modularity*: super-modularity states that the marginal decrease in uncertainty when adding a single query node to an existing set of query nodes decreases as the set becomes larger. Here, we show that super-modularity property does not hold in our setting, even for the simplest case of TREE. In fact, for DAG($2, \wedge$), the problem is hard to approximate within a factor of $O(2^{n^{1-\varepsilon}})$ for any $\varepsilon > 0$. We show that TREE(2) admits a weakly-polynomial exact algorithm and a polynomial approximation algorithm. For general trees, we leave the complexity problem open.

**Picking Nodes Greedily:** First, we show that picking greedily can be arbitrarily bad. Consider a tree with root having $p(r) = 1/2$. There are $2n$ leaves, half with $p = 1$ and rest with $p = 1/2$. If we pick any leaf node with $p = 1$, the expected uncertainty is $n/8$. If we pick a node with $p = 1/2$, the expected uncertainty is $25n/16 - 4/16$. Thus, if we sort nodes by their expected uncertainty, all the $p = 1$ nodes appear before all the $p = 1/2$ nodes. Consider the problem of picking the best $n$ nodes. If we pick greedily based on their expected uncertainty, we pick all the $p = 1$ nodes. However, all of them are perfectly correlated. Thus, expected uncertainty after querying all $p = 1$ nodes is still $n/8$. On the other hand, if we pick a single $p = 1$ node, and $n - 1$ nodes with $p = 1/2$, the resulting uncertainty is a constant. Thus, picking nodes greedily can be $O(n)$ worse than the optimal.

**Counter-example for super-modularity:** Next we show an example from a graph in DAG($2, \wedge$) where super-modularity does not hold. Consider a $G \in$ DAG($2, \wedge$) having two nodes $u$ and $v$ on

the top layer and three nodes $a$, $b$, and $c$ in the bottom layer. Labels of all nodes are $\wedge$. Node $u$ has an edge to $a$ and $b$, while $v$ has an edge to $b$ and $c$. Let $Pr(u) = 1/2$, $Pr(v) = 1/2$, and $Pr(a) = Pr(b) = Pr(c) = 1$. Now consider the expected uncertainty $I_c$ at node $c$. Super-modularity condition implies that $I_c(\{b,a\}) - I_c(\{b\}) \geq I_c(\{a\}) - I_c(\{\})$ (since marginal decrease in expected uncertainty of $c$ on picking an additional node $a$ should be less for set $\{\}$ compared to $\{b\}$). We show that this is violated. First note that $Pr(Y(c)|Y(a))$ is same as $Pr(Y(c))$ (since $Y(a)$ does not affect $Y(v)$ and $Y(c)$). Thus expected uncertainty at $c$ is unaffected by conditioning on $a$ alone, and thus $I_c(\{a\}) = I_c(\{\})$. On the other hand, if $Y(b) = 0$ and $Y(a) = 1$ then $Y(c) = 0$ (since $Y(a) = 1$ implies $Y(u) = 1$ which together with $Y(b) = 0$ implies $Y(v) = 0$ and $Y(c) = 0$). This can be used to show that conditioned on $Y(b)$, expected uncertainty in $c$ drops when conditioning on $Y(a)$. Thus the term $I_c(\{b,a\}) - I_c(\{b\})$ is negative, while we showed that $I_c(\{a\}) - I_c(\{\})$ is 0. This violates the super-modularity condition.

The above example actually shows that super-modularity is violated on $\mathrm{DAG}(\wedge)$ for any choice of metric $f$ in computing expected uncertainty $I$, as long as $f$ is monotonic decreasing away from $1/2$. When $f(p) = p(1-p)$, we can show that super-modularity is violated even for trees as stated in the proposition below.

**Proposition 6.1.** *Let $f(p) = p(1-p)$ be the metric used in computing expected uncertainty $I$. Then there exists a tree $T \in \mathrm{TREE}(d)$ such that for leaf nodes $a$ , $b$, and $c$ in $T$ the following holds: $I_c(\{b,a\}) - I_c(\{b\}) < I_c(\{a\}) - I_c(\{\})$.*

## 6.1 TREE(2)

We now consider the Best-k problem for $\mathrm{TREE}(2)$. As in Section 4, assume the root $r$ with $p(r)$ to be $p_r$, while the leaves $L = \{l_1, \ldots, l_n\}$ have $p(l_i) = p_i$. Let $B = \sum_{l \in L} p^2(l)$. Given a set $Q \subseteq L$, define

$$P(Q) = \prod_{l \in Q} p(l) \qquad S_1(Q) = \sum_{l \in Q} p(l)(1 - p(l)) \qquad S_2(Q) = \sum_{l \in Q} p^2(l)$$

**Lemma 6.2.** *The best set $Q$ of size $k$ is one that minimizes:* $I'(Q) = -S_1(Q) + (B - S_2(Q))\frac{1 - p_r}{(1 - p_r)/P(Q) + p_r}$

(The details of this computation is shown in the extended technical report.) It is easy to check that that the first term is minimized with $Q$ consists of nodes with $p(l)$ closest to $1/2$, and the second term is minimized with nodes with $p(l)$ closest to 1. Intuitively, the first term prefers nodes that are as uncertain as possible, while the second term prefers nodes that reveal as much about the root as possible. This immediately gives us a 2-approximation in the number of queries : by picking at most $2k$ nodes, $k$ closest to $1/2$ and $k$ closest to 1, we can do at least as well as the optimal solution for best-k.

**Exact weakly-polynomial time algorithm:** Note also that as $k$ increases, $P(Q) \rightarrow 0$, and the second term vanishes. This also makes intuitive sense, since the second term prefers nodes that reveal more about the root, and once we use sufficiently many nodes to infer the correctness of the root, we do not get any gain from asking additional questions. Thus, we set a constant $c_\tau$, depending on the $p_i$, such that if $k < c_\tau$, we consider all possible choices of $k$ queries, and if $k \geq c_\tau$, we may simply pick the $k$ largest $p_i$, because the second term would be very small. We describe this algorithm along with the proof in the extended technical report [11].

## 6.2 DAG($\wedge$):

**Theorem 6.3** (PP-Hardness of Incr. for $\mathrm{DAG}(\wedge)$)**.** *The best-k problem in $\mathrm{DAG}(\wedge)$ is PP-Hard.*

The proof can be found in the extended technical report [11]. Our reduction is a weakly parsimonious reduction involving monotone-partitioned-2-CNF, which is known to be hard to approximate, thus we have the following result:

**Theorem 6.4** (Inapproximability for $\mathrm{DAG}(\wedge)$)**.** *The best-k problem for $\mathrm{DAG}(\wedge)$ is hard to approximate.*

# 7 Conclusion

In this work, we performed a detailed complexity analysis for the problem of finding optimal set of query nodes for various classes of graphs. We showed that for trees, most of the problems are tractable, and in fact quite efficient. For general dags, they become hard to even approximate. We leave open the complexity of the best-k problem for trees.

# References

[1] Ashraf M. Abdelbar and Sandra M. Hedetniemi. Approximating maps for belief networks is np-hard and other theorems. *Artif. Intell.*, 102(1):21–38, June 1998.

[2] Maria-Florina Balcan, Alina Beygelzimer, and John Langford. Agnostic active learning. *J. Comput. Syst. Sci.*, 75(1):78–89, 2009.

[3] Kedar Bellare, Suresh Iyengar, Aditya Parameswaran, and Vibhor Rastogi. Active sampling for entity matching. In *KDD*, 2012.

[4] Alina Beygelzimer, Sanjoy Dasgupta, and John Langford. Importance weighted active learning. In *ICML*, page 7, 2009.

[5] Alina Beygelzimer, Daniel Hsu, John Langford, and Tong Zhang. Agnostic active learning without constraints. In *NIPS*, pages 199–207, 2010.

[6] Christopher M. Bishop. *Pattern Recognition and Machine Learning (Information Science and Statistics).* Springer, 1 edition, 2007.

[7] Philip Bohannon, Srujana Merugu, Cong Yu, Vipul Agarwal, Pedro DeRose, Arun Iyer, Ankur Jain, Vinay Kakade, Mridul Muralidharan, Raghu Ramakrishnan, and Warren Shen. Purple sox extraction management system. *SIGMOD Rec.*, 37:21–27, March 2009.

[8] Gregory F. Cooper. The computational complexity of probabilistic inference using bayesian belief networks. *Artif. Intell.*, 42(2-3):393–405, 1990.

[9] Nilesh Dalvi, Ravi Kumar, Bo Pang, Raghu Ramakrishnan, Andrew Tomkins, Philip Bohannon, Sathiya Keerthi, and Srujana Merugu. A web of concepts (keynote). In *PODS*. Providence, Rhode Island, USA, June 2009.

[10] Nilesh Dalvi, Ravi Kumar, and Mohamed A. Soliman. Automatic wrappers for large scale web extraction. *PVLDB*, 4(4):219–230, 2011.

[11] Nilesh Dalvi, Aditya Parameswaran, and Vibhor Rastogi. Minimizing uncertainty in pipelines. Technical report, Stanford Infolab, 2012.

[12] Sanjoy Dasgupta and John Langford. Tutorial summary: Active learning. In *ICML*, page 178, 2009.

[13] Yoav Freund, H. Sebastian Seung, Eli Shamir, and Naftali Tishby. Selective sampling using the query by committee algorithm. *Machine Learning*, 28(2-3):133–168, 1997.

[14] Pankaj Gulhane, Rajeev Rastogi, Srinivasan H. Sengamedu, and Ashwin Tengli. Exploiting content redundancy for web information extraction. *PVLDB*, 3(1):578–587, 2010.

[15] Steve Hanneke. A bound on the label complexity of agnostic active learning. In *ICML*, pages 353–360, 2007.

[16] Don Herbison-Evans. Solving quartics and cubics for graphics. 1994.

[17] Nikos Karampatziakis and John Langford. Online importance weight aware updates. In *UAI*, pages 392–399, 2011.

[18] Richard M. Karp and Michael Luby. Monte-carlo algorithms for enumeration and reliability problems. In *Proceedings of the 24th Annual Symposium on Foundations of Computer Science*, pages 56–64, 1983.

[19] Andreas Krause and Carlos Guestrin. Near-optimal nonmyopic value of information in graphical models. In *UAI*, pages 324–331, 2005.

[20] Andreas Krause and Carlos Guestrin. Near-optimal observation selection using submodular functions. In *AAAI*, pages 1650–1654, 2007.

[21] J. Kwisthout. The Computational Complexity of Probabilistic Inference. Technical Report ICIS–R11003, Radboud University Nijmegen, April 2011.

[22] Michael L. Littman, Stephen M. Majercik, and Toniann Pitassi. Stochastic boolean satisfiability. *J. Autom. Reasoning*, 27(3):251–296, 2001.

[23] A. Parameswaran, A. Das Sarma, H. Garcia-Molina, N. Polyzotis, and J. Widom. Human-assisted graph search: it's okay to ask questions. In *VLDB*, 2011.

[24] James D. Park and Adnan Darwiche. Complexity results and approximation strategies for map explanations. *J. Artif. Intell. Res. (JAIR)*, 21:101–133, 2004.

[25] J. Scott Provan and Michael O. Ball. The complexity of counting cuts and of computing the probability that a graph is connected. *SIAM J. Comput.*, 12(4):777–788, 1983.

[26] Dan Roth. On the hardness of approximate reasoning. *Artif. Intell.*, 82(1-2):273–302, 1996.

[27] Burr Settles. Active learning literature survey. Computer Sciences Technical Report 1648, University of Wisconsin–Madison, 2009.

[28] Alice X. Zheng, Irina Rish, and Alina Beygelzimer. Efficient test selection in active diagnosis via entropy approximation. In *UAI*, pages 675–, 2005.

